# Static Analysis of Binary Executables Using Structural SVMs

**Nikos Karampatziakis**[*]
Department of Computer Science
Cornell University
Ithaca, NY 14853
nk@cs.cornell.edu

## Abstract

We cast the problem of identifying basic blocks of code in a binary executable as learning a mapping from a byte sequence to a segmentation of the sequence. In general, inference in segmentation models, such as semi-CRFs, can be cubic in the length of the sequence. By taking advantage of the structure of our problem, we derive a linear-time inference algorithm which makes our approach practical, given that even small programs are tens or hundreds of thousands bytes long. Furthermore, we introduce two loss functions which are appropriate for our problem and show how to use structural SVMs to optimize the learned mapping for these losses. Finally, we present experimental results that demonstrate the advantages of our method against a strong baseline.

## 1   Introduction

In this work, we are interested in the problem of extracting the CPU instructions that comprise a binary executable file. Solving this problem is an important step towards verifying many simple properties of a given program. In particular we are motivated by a computer security application, in which we want to detect whether a previously unseen executable contains malicious code. This is a task that computer security experts have to solve many times every day because in the last few years the volume of malicious software has witnessed an exponential increase (estimated at 50000 new malicious code samples every day). However, the tools that analyze binary executables require a lot of manual effort in order to produce a correct analysis. This happens because the tools themselves are based on heuristics and make many assumptions about the way a binary executable is structured.

But why is it hard to find the instructions inside a binary executable? After all, when running a program the CPU always knows which instructions it is executing. The caveat here is that we want to extract the instructions from the executable without running it. On one hand, running the executable will in general reveal little information about all possible instructions in the program, and on the other hand it may be dangerous or even misguiding.[1]

Another issue that makes this task challenging is that binary executables contain many other things except the instructions they will execute.[2] Furthermore, the executable does not contain any demarcations about the locations of instructions in the file.[3] Nevertheless, an executable file is organized into sections such as a code section, a section with constants, a section containing global variables etc. But even inside the code section, there is a lot more than just a stream of instructions. We will

---

[*]http://www.cs.cornell.edu/~nk

[1]Many malicious programs try to detect whether they are running under a controlled environment.

[2]Here, we are focusing on Windows executables for the Intel x86 architecture, though everything carries over to any other modern operating system and any other architecture with a complex instruction set.

[3]Executables that contain debugging information are an exception, but most software is released without it

refer to all instructions as *code* and to everything else as *data*. For example, the compiler may, for performance reasons, prefer to pad a function with up to 3 data bytes so that the next function starts at an address that is a multiple of 4. Moreover, data can appear inside functions too. For example, a "switch" statement in C is usually implemented in assembly using a table of addresses, one for each "case" statement. This table does not contain any instructions, yet it can be stored together with the instructions that make up the function in which the "switch" statement appears. Apart from the compiler, the author of a malicious program can also insert data bytes in the code section of her program. The ultimate goal of this act is to confuse the heuristic tools via creative uses of data bytes.

## 1.1 A text analogy

To convey more intuition about the difficulties in our task we will use a text analogy. The following is an excerpt from a message sent to General Burgoyne during the American revolutionary war [1]:

> You will have heard, Dr Sir I doubt not long before this can have reached you that **Sir W. Howe** is gone from hence. The Rebels imagine that he **is gone to the** Eastward. By this time however he has filled **Chesapeak bay with** surprize and terror. Washington marched **the greater part of the** Rebels to Philadelphia in order to oppose Sir Wm's. **army**.

The sender also sent a mask via a different route that, when placed on top of the message, revealed only the words that are shown here in bold. Our task can be thought as learning what needs to be masked so that the hidden message is revealed. In this sense, words play the role of instructions and letters play the role of bytes. For complex instruction sets like the Intel x86, instructions are composed of a variable number of bytes, as words are composed of a variable number of letters. There are also some minor differences. For example, programs have control logic (i.e. execution can jump from one point to another), while text is read sequentially. Moreover, programs do not have spaces while most written languages do (exceptions are Chinese, Japanese, and Thai).

This analogy motivates tackling our problem as predicting a segmentation of the input sequence into blocks of code and blocks of data. An obvious first approach for this task would be to treat it as a sequence labeling problem and train, for example, a linear chain conditional random field (CRF) [2] to tag each byte in the sequence as being the beginning, inside, or outside of a data block. However this approach ignores much of the problem's structure, most importantly that transitions from code to data can only occur at specific points. Instead, we will use a more flexible model which, in addition to sequence labeling features, can express features of whole code blocks. Inference in our model is as fast as for sequence labeling and we show a connection to weighted interval scheduling. This strikes a balance between efficient but simple sequence labeling models such as linear chain CRFs, and expressive but slow[4] segmentation models such as semi-CRFs [3] and semi-Markov SVMs [4]. To learn the parameters of the model, we will use structural SVMs to optimize performance according to loss functions that are appropriate for our task, such as the sum of incorrect plus missed CPU instructions induced by the segmentation.

Before explaining our model in detail, we present some background on the workings of widely used tools for binary code analysis in section 2, which allows us to easily explain our approach in section 3. We empirically demonstrate the effectiveness of our model in section 4 and discuss related work and other applications in section 5. Finally, section 6 discusses future work and states our conclusions.

## 2 Heuristic tools for analyzing binary executables

Tools for statically analyzing binary executables differ in the details of their workings but they all share the same high level logic, which is called *recursive disassembly*.[5] The tool starts by obtaining the address of the first instruction from a specific location inside the executable. It then places this address on a stack and executes the following steps while the stack is non-empty. It takes the next

address from the stack and *disassembles* (i.e. decompiles to assembly) the sequence starting from that address. All the disassembled instructions would execute one after the other until we reach an instruction that changes the flow of execution. These *control flow* instructions, are conditional and unconditional jumps, calls, and returns. After the execution of an unconditional jump the next instruction to be executed is at the address specified by the jump's argument. Other control flow instructions are similar to the unconditional jump. A conditional jump also specifies a condition and does nothing if the condition is false. A call saves the address of the next instruction and then jumps. A return jumps to the address saved by a call (and does not need an address as an argument). The tool places the arguments of control flow instructions it encounters on the stack. If the control flow instruction is a conditional jump or a call, it continues disassembling, otherwise it takes the next address, that has not yet been disassembled, from the stack and repeats.

Even though recursive disassembly seems like a robust way of extracting the instructions from a program, there are many reasons that can make it fail [6]. Most importantly, the arguments of the control flow instructions do not have to be constants, they can be registers whose values are generally not available during static analysis. Hence, recursive disassembly can ran out of addresses much before all the instructions have been extracted. After this point, the tool has to resort to heuristics to populate its stack. For example, a heuristic might check for positions in the sequence that match a hand-crafted regular expression. Furthermore, some heuristics have to be applied on multiple passes over the sequence. According to its documentation, OllyDbg does 12 passes over the sequence.

Recursive disassembly can also fail because of its assumptions. Recall that after encountering a call instruction, it continues disassembling the next instruction, assuming that the call will eventually return to execute it. Similarly for a conditional jump it assumes that both branches can potentially execute. Though these assumptions are reasonable for most programs, malicious programs can exploit them to confuse the static analysis tools. For example, the author of a malicious program can write a function that, say, adds 3 to the return address that was saved by the call instruction. This means that if the call instruction was spanning positions $a, \ldots, a + \ell - 1$ of the sequence, upon the function's return the next instruction will be at position $a + \ell + 3$, not at $a + \ell$. This will give a completely different decoding of the sequence and is called *disassembly desynchronization*. To return to a text analogy, recursive disassembly parses the sequence "driverballetterrace" as [driver, ballet, terrace] while the actual parsing, obtained by starting three positions down, is [xxx, verbal, letter, race], where x denotes junk data.

## 3   A structured prediction model

In this section we will combine ideas from recursive disassembly and structured prediction to derive an expressive and efficient model for predicting the instructions inside a binary executable. As in recursive disassembly, if we are certain that code begins at position $i$ we can unambiguously disassemble the byte sequence starting from position $i$ until we reach a control flow instruction. But unlike recursive disassembly, we maintain a trellis graph, a directed graph that succinctly represents all possibilities. The trellis graph has vertices $b_i$ that denote the possibility that a code block starts at position $i$. It also has vertices $e_j$ and edges $(b_i, e_j)$ which denote that disassembling from position $i$ yields a possible code block that spans positions $i, \ldots, j$. Furthermore, vertices $d_i$ denote the possibility that the $i$-th position is part of a data block. Edges $(e_j, b_{j+1})$ and $(e_j, d_{j+1})$ encode that the next byte after a code block can either be the beginning of another code block, or a data byte respectively. For data blocks no particular structure is assumed and we just use edges $(d_j, d_{j+1})$ and $(d_j, b_{j+1})$ to denote that a data byte can be followed either by another data byte or by the beginning of a code block respectively. Finally, we include vertices $s$ and $t$ and edges $(s, b_1)$, $(s, d_1)$, $(d_n, t)$ and $(e_n, t)$ to encode that sequences can start and end either with code or data.

An example is shown in Figure 1. The graph encodes all possible valid[6] segmentations of the sequence. In fact, there is a simple bijection $P$ from any valid segmentation $y$ to an $s - t$ path $P(y)$ in this graph. For example, the sequence in Figure 1 contains three code blocks that span positions 1–7, 8–8, and 10–12. This segmentation can be encoded by the path $s, b_1, e_7, b_8, e_8, d_9, b_{10}, e_{12}, t$.

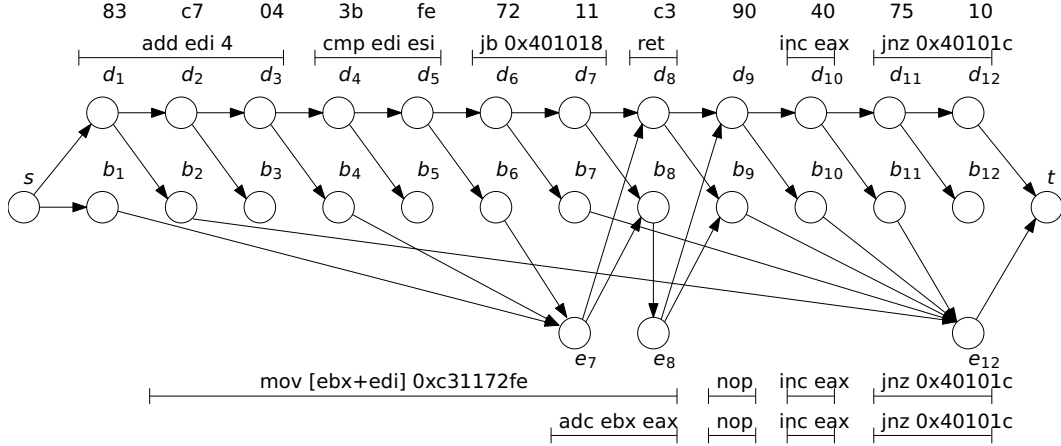

Figure 1: The top line shows an example byte sequence in hexadecimal. Below this, we show the actual x86 instructions with position 9 being a data byte. We show both the mnemonic instructions as well as the bytes they are composed of. Some alternative decodings of the sequence are shown on the bottom. The decoding that starts from the second position is able to skip over two control flow instructions. In the middle we show the graph that captures all possible decodings of the sequence. Disassembling from positions 3, 5, and 12 leads to decoding errors.

As usual for predicting structured outputs [2] [7], we define the score of a segmentation $y$ for a sequence $x$ to be the inner product $w^\top \Psi(x, y)$ where $w \in \mathbb{R}^d$ are the parameters of our model and $\Psi(x, y) \in \mathbb{R}^d$ is a vector of features that captures the compatibility of the segmentation $y$ and the sequence $x$. Given a sequence $x$ and a vector of parameters $w$, the inference task is to find the highest scoring segmentation

$$\hat{y} = \underset{y \in \mathcal{Y}}{\operatorname{argmax}} \, w^\top \Psi(x, y) \tag{1}$$

where $\mathcal{Y}$ is the space of all valid segmentations of $x$. We will assume that $\Psi(x, y)$ decomposes as

$$\Psi(x, y) := \sum_{(u,v) \in P(y)} \Phi(u, v, x)$$

where $\Phi(u, v, x)$ is a vector of features that can be computed using only the endpoints of edge $(u, v)$ and the byte sequence. This assumption allows efficient inference because (1) can be rewritten as

$$\hat{y} = \underset{y \in \mathcal{Y}}{\operatorname{argmax}} \sum_{(u,v) \in P(y)} w^\top \Phi(u, v, x)$$

which we recognize as computing the heaviest path in the trellis graph with edge weights given by $w^\top \Phi(u, v, x)$. This problem can be solved efficiently with dynamic programming by visiting each vertex in topological order and updating the longest path to each of its neighbors.

The inference task can be viewed as a version of the weighted interval scheduling problem. Disassembling from position $i$ in the sequence yields an interval $[i, j]$ where $j$ is the position where the first encountered control flow instruction ends. In weighted interval scheduling we want to select a subset of non-overlapping intervals with maximum total weight. Our inference problem is the same except we also have a cost for switching to the next interval, say the one that starts at position $j + 2$, which is captured by the cost of the path $e_j, d_{j+1}, b_{j+2}$. Finally, the dynamic programming algorithm for solving this version is a simple modification of the classic weighted interval scheduling algorithm. Section 5 discusses other setups where this inference problem arises.

### 3.1 Loss functions

Now we introduce loss functions that measure how close an inferred segmentation $\hat{y}$ is to the real one $y$. First, we argue that Hamming loss, how well the bytes of the blocks in $\hat{y}$ overlap with with the bytes of the blocks in $y$, is not appropriate for this task because, as we recall from the text

analogy at the end of section 2, two blocks may be overlapping very well but they may lead to completely different decodings of the sequence. Hence, we introduce two loss functions which are more appropriate for our task.

The first loss function, which we call block loss, comes from the observation that the beginnings of the code blocks are necessary and sufficient to describe the segmentation. Therefore, we let $y$ and $\hat{y}$ be the sets of positions where the code blocks start in the two segmentations and the block loss counts how well these two sets overlap using the cardinality of their symmetric difference

$$\Delta_B(y, \hat{y}) = |y| + |\hat{y}| - 2|y \cap \hat{y}|$$

The second loss function, which we call instruction loss, is a little less stringent. In the case where the inferred $\hat{y}$ identifies, say, the second instruction in a block as its start, we would like to penalize this less, since the disassembly is still synchronized, and only missed one instruction. Formally, we let $y$ and $\hat{y}$ be the sets of positions where the instructions start in the two segmentations and we define the instruction loss $\Delta_I(y, \hat{y})$ to be the cardinality of their symmetric difference.

As an example, consider the segmentation which corresponds to path $s, d_1, b_2, e_{12}, t$ in Figure 1. Therefore $\hat{y} = \{2\}$ and from the figure we see that the segmentation in the top line has to pass through $b_1, b_8, b_{10}$ i.e. $y = \{1, 8, 10\}$. Hence its block loss is 4 because it misses $b_1, b_8, b_{10}$ and it introduces $b_2$. For the instruction loss, the positions of the real instructions are $y = \{1, 4, 6, 8, 10, 11\}$ while the proposed segmentation has $\hat{y} = \{2, 9, 10, 11\}$. Taking the symmetric difference of these sets, we see that the instruction loss has value 6.

Finally a variation of these loss functions occurs when we aggregate the losses over a set of sequences. If we simply sum the losses for each sequence then the losses in longer executables may overshadow the losses on shorter ones. To represent each executable equally in the final measure we can normalize our loss functions, for example we can define the normalized instruction loss to be

$$\Delta_{NI}(y, \hat{y}) = \frac{|y| + |\hat{y}| - 2|y \cap \hat{y}|}{|y|}$$

and we similarly define a normalized block loss $\Delta_{NB}$. If $|\hat{y}| = |y|$, $\Delta_{NI}$ and $\Delta_{NB}$ are scaled versions of a popular loss function $1 - F_1$, where $F_1$ is the harmonic mean of precision and recall.

### 3.2 Training

Given a set of training pairs $(x_i, y_i)$ $i = 1, \ldots, n$ of sequences and segmentations we can learn a vector of parameters $w$, that assigns a high score to segmentation $y_i$ and a low score to all other possible segmentations of $x_i$. For this we will use the structural SVM formulation with margin rescaling [7] that solves the following problem:

$$\min_{w, \xi_i} \frac{1}{2}||w||^2 + \frac{C}{n} \sum_{i=1}^{n} \xi_i$$

$$\text{s.t. } \forall i \, \forall \bar{y} \in \mathcal{Y}_i : \quad w^\top \Psi(x_i, y_i) - w^\top \Psi(x_i, \bar{y}) \geq \Delta(y_i, \bar{y}) - \xi_i$$

The constraints of this optimization problem enforce that the difference in score between the correct segmentation $y$ and any incorrect segmentation $\bar{y}$ is at least as large as the loss $\Delta(y_i, \bar{y})$. If $\hat{y}_i$ is the inferred segmentation then the slack variable $\xi_i$ upper bounds $\Delta(y_i, \hat{y}_i)$. Hence, the objective is a tradeoff between a small upper bound of the average training loss and a low-complexity hypothesis $w$. The tradeoff is controlled by $C$ which is set using cross-validation. Since the sets of valid segmentations $\mathcal{Y}_i$ are exponentially large, we solve the optimization problem with a cutting plane algorithm [7]. We start with an empty set of constraints and in each iteration we find the most violated constraint for each example. We add these constraints in our optimization problem and re-optimize. We do this until there are no constraints which are violated by more than a prespecified tolerance $\epsilon$. This procedure will terminate after $O(\frac{1}{\epsilon})$ iterations [8]. For a training pair $(x_i, y_i)$ the most violated constraint is:

$$\hat{y} = \underset{\bar{y} \in \mathcal{Y}_i}{\operatorname{argmax}} \, w^\top \Psi(x_i, \bar{y}) + \Delta(y_i, \bar{y}) \tag{2}$$

Apart from the addition of $\Delta(y_i, \bar{y})$, this is the same as the inference problem. For the losses we introduced, we can solve the above problem with the same inference algorithm in a slightly modified

| | Bytes | Blocks | Block length (bytes) | Block length (instructions) |
|---|---|---|---|---|
| Maximum | 49152 | 3502 | 2794 | 1009 |
| Average | 16712 | 887 | 13 | 4 |

Table 1: Some statistics about the executable sections of the programs in the dataset

trellis graph. More precisely, for every vertex $v$ we can define a cost $c(v)$ for visiting it (this can be absorbed into the costs of $v$'s incoming edges) and find the longest path in this modified graph. This is possible because our losses decompose over the vertices of the graph. This is not true for losses such $1 - F_1$ for which (2) seems to require time quadratic in the length of the sequence.

For the block loss, the costs are defined as follows. If $b_i \in y$ then $c(d_i) = 1$. This encodes that using $d_i$ instead of $b_i$ misses the beginning of one block. If $b_i \notin y$ then $b_i$ defines an incorrect code block which spans bytes $i, \dots, j$ and $c(b_i) = 1 + |\{k | b_k \in y \wedge i < k \leq j\}|$, capturing that we will introduce one incorrect block and we will skip all the blocks that begin between positions $i$ and $j$. All other vertices in the graph have zero cost. In Figure 1 vertices $d_1$, $d_8$ and $d_{10}$ have a cost of $1$, while $b_2$, $b_4$, $b_6$, $b_7$, $b_9$, and $b_{11}$ have costs $3, 1, 1, 3, 2$, and $1$ respectively.

For the instruction loss, $y$ is a set of instruction positions. Similarly to the block loss if $i \in y$ then $c(d_i) = 1$. If $i \notin y$ then $b_i$ is the beginning of an incorrect block that spans bytes $i, \dots, j$ and produces instructions in a set of positions $\tilde{y}_i$. Let $s$ be the first position in this block that gets synchronized with the correct decoding i.e. $s = \min(\tilde{y}_i \cap y)$ with $s = j$ if the intersection is empty. Then $c(b_i) = |\{k | k \in \tilde{y}_i \wedge i \leq k < s\}| + |\{k | k \in y \wedge i < k < s\}|$. The first term captures the number of incorrect instructions produced by treating $b_i$ as the start of a code block, while the second term captures the number of missed real instructions. All other vertices in the graph have zero cost. In Figure 1 vertices $d_1$, $d_4$, $d_6$, $d_8$, $d_{10}$ and $d_{11}$ have a cost of $1$, while $b_2$, $b_7$, and $b_9$ have costs $5, 3$, and $1$ respectively. For the normalized losses, we simply divide the costs by $|y|$.

## 4 Experiments

To evaluate our model we tried two different ways of collecting data, since we could not find a publicly available set of programs together with their segmentations. First, we tried using debugging information, i.e. compile a program with and without debugging information and use the debug annotations to identify the code blocks. This approach could not discover all code blocks, especially when the compiler was automatically inserting code that did not exist in the source, such as the calls to destructors generated by C++ compilers. Therefore we resorted to treating the output of OllyDbg, a heuristic tool, as the ground truth. Since the executables we used were 200 common programs from a typical installation of Windows XP, we believe that the outputs of heuristic tools should have little noise. For a handful of programs we manually verified that another heuristic tool, IdaPro, mostly agreed with OllyDbg. Of course, our model is a general statistical model and given an expressive feature map, it can learn any ground truth. In this view the experiments suggest the relative performance of the compared models. The dataset, and an implementation of our model, is available at http://www.cs.cornell.edu/~nk/svmwis. Table 1 shows some statistics of the dataset.

We use two kinds of features, byte-level and instruction-level features. For each edge in the graph, the byte-level features are extracted from an 11 byte window around the source of the edge (so if the source vertex is at position $i$, the window spans positions $i - 5, \dots, i + 5$). The features are which bytes and byte pairs appear in which position inside the window. An example feature is "does byte c3 appear in position $i - 1$?". In x86 architectures, when the previous instruction is a return instruction this feature fires. Of course, it also fires in other cases and that is why we need instruction-level features. These are obtained from histograms of instructions that occur in candidate code blocks (i.e. edges of the form $(b_i, e_j)$). We use two kinds of histograms, one where we abstract the values of the arguments of the instructions but keep their type (register, memory location or constant), and one where we completely discard all information about the arguments. An example of the former type of feature would be "number of times the instruction [add register, register] appears in this block". An example of the latter type of feature would be "number of times the instruction [mov] appears in this block". In total, we have 2.3 million features. Finally, we normalize the features by dividing them by the length of the sequence.

|  | $\Delta_H$ | $\bar{L} \cdot \Delta_{NH}$ | $\Delta_I$ | $\bar{I} \cdot \Delta_{NI}$ | $\Delta_B$ | $\bar{B} \cdot \Delta_{NB}$ |
|---|---|---|---|---|---|---|
| Greedy | 1623.6 | 1916.6 | 2164.3 | 7045.2 | 1564.9 | 4747.2 |
| SVM$^{hmm}$ | 236.2 | 201.3 | — | — | 45.1 | 46.9 |
| SVM$^{wis}$ $\Delta_I$ | 98.8 | 115.6 | 44.6 | 98.0 | 26.1 | 41.1 |
| SVM$^{wis}$ $\Delta_{NI}$ | 104.3 | 103.7 | 45.5 | 79.7 | 30.5 | 35.5 |
| SVM$^{wis}$ $\Delta_B$ | 86.5 | 98.2 | **39.6** | 80.2 | **21.5** | 32.1 |
| SVM$^{wis}$ $\Delta_{NB}$ | **85.2** | **87.2** | 40.6 | **75.4** | 23.4 | **29.8** |

Table 2: Empirical results. $\Delta_H$ is Hamming loss. Normalized losses ($\Delta_{NX}$) are multiplied with the average number of bytes ($\bar{L}$), instructions ($\bar{I}$), or blocks ($\bar{B}$) to bring all numbers to a similar scale.

We compare our model SVM$^{wis}$ (standing for weighted interval scheduling, to underscore that it is not a general segmentation model), trained to minimize the losses we introduced, with a very strong baseline, a discriminatively trained HMM (using SVM$^{hmm}$). This model uses only the byte-level features since it cannot express the instruction-level features. It tags each byte as being the beginning, inside or outside of a code block using Viterbi and optimizes Hamming loss. Running a general segmentation model [4] was impractical since inference depends quadratically on the maximum length of the code blocks, which was 2800 in our data. Finally, it would be interesting to compare with [5], but we could not find their inference algorithm available as a ready to use software. For all experiments we use five fold cross-validation where three folds are used for training one fold for validation (selecting $C$) and one fold for testing.

Table 2 shows the results of our comparison for different loss functions (columns): Hamming loss, instruction loss, block loss, and their normalized counterparts. Results for normalized losses have been multiplied with the average number of bytes ($\bar{L}$), instructions ($\bar{I}$), or blocks ($\bar{B}$) to bring all numbers to a similar scale. To highlight the stregth of our main baseline, SVM$^{hmm}$, we have included a very simple baseline which we call greedy. Greedy starts decoding from the begining of the sequence and after decoding a block ($b_i, e_j$) it repeats at position $j + 1$. It only marks a byte as data if the decoding fails, in which case it starts decoding from the next byte in the sequence. The results suggest that just treating our task as a simple sequence labeling problem at the level of bytes already goes a long way in terms of Hamming loss and block loss. SVM$^{hmm}$ sometimes predicts as the beginning of a code block a position that leads to a decoding error. Since it is not clear how to compute the instruction loss in this case, we do not report instruction losses for this model. The last four rows of the table show the results for our model, trained to minimize the loss indicated on each line. We observe a further reduction in loss for all of our models. To assess this reduction, we used paired Wilcoxon signed rank tests between the losses of SVM$^{hmm}$'s predictions and the losses of our model's predictions (200 pairs). For all four models the tests suggest a statistically significant improvement over SVM$^{hmm}$ at the 1% level. For the block loss and its normalized version $\Delta_{NB}$, we see that the best performance is obtained for the model trained to minimize the respective loss. However this is not true for the other loss functions. For the Hamming loss, this is expected since the SVM$^{wis}$ models are more expressive and a small block loss or instruction loss implies a small Hamming loss, but not vice versa. For the instruction loss, we believe this occurs because of two reasons. First our data consists of benign programs and for them learning to identify the code blocks may be enough. Second it may be harder to learn with the instruction loss since its value depends on how quickly each decoding synchronizes with another (the correct) decoding of the stream, something that is not modeled in the feature map we are using. The end result is that the models trained for block loss also attain the smallest losses for all other loss functions.

## 5   Related work and other applications

There are two lines of research which are relevant to this work: one is structured prediction approaches for segmenting sequences and the other is research on static analysis techniques for finding code and data blocks in executables. Segmentation of sequences can be done via sequence labeling e.g. [9]. If features of whole segments are needed then more expressive models such as semi-CRFs [3] or semi-Markov SVMs [4] can be used. The latter work introduced training of segmentation models for specific losses. However, if the segments are allowed to be long enough, these models have polynomial but impractical inference complexity. With additional assumptions on the features

[5] gives an efficient, though somewhat complicated, inference algorithm. In our model inference takes linear time, is simple to implement, and does not depend on the length of the segments.

Previous techniques for identifying code blocks in executables have used no or very little statistical learning. For example, [10] and [11] use recursive disassembly and pattern heuristics similarly to currently used tools such as OllyDbg and IdaPro. These heuristics make many assumptions about the data which are lucidly explained in [6]. In this work, the authors use simple statistical models based on unigram and bigram instruction models in addition to the pattern heuristics. However, these approaches make independent decisions for every candidate code block and they have a less principled way of dealing with equally plausible but overlapping code blocks.

Our work is most similar to [12] which uses a CRF to locate the entry points of functions. They use features that induce pairwise interactions between all possible positions in the executable which makes their formulation intractable. They perform approximate inference with a custom iterative algorithm but this is still slow. Our model can capture all the types of features that were used in that model except one. This feature encodes whether an address that is called by a function is not marked as a function and including this in our structure would make exact inference NP-hard. One way to approximate this feature would be to count how many candidate code blocks have instructions that jump to or call the current position in the sequence. For their task, compiling with debugging information was enough to get real labels and they showed that, according to these labels, heuristic tools are outperformed by their learning approach.

Finally, we conclude this section with a discussion on the broader impact of this work. Our model is a general structured learning model and can be used in many sequence labeling problems. First, it can encode all features of a linear chain CRF and can simulate it by specifying a structure where each block is required to end at the same position where it starts. Furthermore, it can be used for any application where each position can yield at most one or a small number of *arbitrarily long* possible intervals and still have linear time inference, while inference in segmentation models depends on the length of the segments. Applications of this form can arise in any kind of scheduling problem where we want to learn a scheduler from example schedules. For example, a news website may decide to show an ad in their front page together with their news stories. Each advertiser submits an ad along with the times on which they want the ad to be shown. The news website can train a model like the one we proposed based on past schedules and the observed total profit for each of those days. The profit may not be directly observable for each individual ad depending on who serves the ads. When one or more ads change in the future, the model can still create a good schedule because its decisions depend on the features of the ads (such as the words in each ad), the time selected for displaying the ad as well as the surrounding ads.

# 6 Conclusions

In this work we proposed a code segmentation model SVM$^{wis}$ that can help security experts in the static analysis of binary executables. We showed that inference in this model is as fast as for sequence labeling, even though our model can have features that can be computed from entire blocks of code. Moreover, our model is trained for the loss functions that are appropriate for the task. We also compared our model with a very strong baseline, a sequence labeling approach using a discriminatively trained HMM, and showed that we consistently outperform it.

In the future we would like to use data annotated with real segmentations which might be possible to extract via a closer look at the compilation and linking process. We also want to look into richer features such as some approximation of call consistency (since the actual constraints give rise to NP-hard inference), so that addresses which are targets of call or jump instructions from a code block do not lie inside data blocks. Finally, we plan to extend our model to allow for joint segmentation and classification of the executable as malicious or not.

**Acknowledgments**

I would like to thank Adam Siepel for bringing segmentation models to my attention and Thorsten Joachims, Dexter Kozen, Ainur Yessenalina, Chun-Nam Yu, and Yisong Yue for helpful discussions.

## Footnotes

[4]More specifically, inference needs $O(nL^2)$ time where $L$ is an a priori bound on the lengths of the segments ($L = 2800$ in our data) and $n$ is the length of the sequence. With additional assumptions on the features, [5] gives an $O(nM)$ algorithm where $M$ is the maximum span of any edge in the CRF.

[5]Two example tools are IdaPro (http://www.hex-rays.com/idapro) and OllyDbg (http://www.ollydbg.de)

[6]Some subsequences will produce errors while decoding to assembly because some bytes may not correspond to any instructions. These could never be valid code blocks because they would crash the program. Also the program cannot do something interesting and crash in the same code block; interesting things can only happen with system calls which, being call instructions, have to be at the end of their code block

# References

[1] F. B. Wrixon Codes, Ciphers, Secrets and Cryptic Communication. page 490, Black Dog & Leventhal Publishers, 2005.

[2] John D. Lafferty, Andrew McCallum, and Fernando C. N. Pereira. Conditional random fields: Probabilistic models for segmenting and labeling sequence data. In *ICML '01: Proceedings of the Eighteenth International Conference on Machine Learning*, pages 282–289, San Francisco, CA, USA, 2001. Morgan Kaufmann Publishers Inc.

[3] S. Sarawagi and W.W. Cohen. Semi-markov conditional random fields for information extraction. *Advances in Neural Information Processing Systems*, 17:1185–1192, 2005.

[4] Q. Shi, Y. Altun, A. Smola, and SVN Vishwanathan. Semi-Markov Models for Sequence Segmentation. In *Proceedings of the 2007 EMNLP-CoNLL*.

[5] S. Sarawagi. Efficient inference on sequence segmentation models. In *Proceedings of the 23rd international conference on Machine learning*, page 800. ACM, 2006.

[6] C. Kruegel, W. Robertson, F. Valeur, and G. Vigna. Static disassembly of obfuscated binaries. In *Proceedings of the 13th conference on USENIX Security Symposium-Volume 13*, page 18. USENIX Association, 2004.

[7] I. Tsochantaridis, T. Joachims, T. Hofmann, and Y. Altun. Large margin methods for structured and interdependent output variables. *Journal of Machine Learning Research*, 6(2):1453, 2006.

[8] T. Joachims, T. Finley, and C-N. Yu. Cutting-Plane Training of Structural SVMs. *Machine Learning*, 77(1):27, 2009.

[9] F. Sha and F. Pereira. Shallow parsing with conditional random fields. In *Proceedings of HLT-NAACL*, pages 213–220, 2003.

[10] H. Theiling. Extracting safe and precise control flow from binaries. In *Seventh International Conference on Real-Time Computing Systems and Applications*, pages 23–30, 2000.

[11] C. Cifuentes and M. Van Emmerik. UQBT: Adaptable binary translation at low cost. *Computer*, 33(3):60–66, 2000.

[12] N. Rosenblum, X. Zhu, B. Miller, and K. Hunt. Learning to analyze binary computer code. In *Conference on Artificial Intelligence (AAAI 2008), Chicago, Illinois*, 2008.

